# AN ADAPTIVE AND HETERODYNE FILTERING PROCEDURE FOR THE IMAGING OF MOVING OBJECTS

F. H. Schuling, H. A. K. Mastebroek and W. H. Zaagman
Biophysics Department, Laboratory for General Physics
Westersingel 34, 9718 CM Groningen, The Netherlands

## ABSTRACT

Recent experimental work on the stimulus velocity dependent time resolving power of the neural units, situated in the highest order optic ganglion of the blowfly, revealed the at first sight amazing phenomenon that at this high level of the fly visual system, the time constants of these units which are involved in the processing of neural activity evoked by moving objects, are -roughly spoken- inverse proportional to the velocity of those objects over an extremely wide range. In this paper we will discuss the implementation of a two dimensional heterodyne adaptive filter construction into a computer simulation model. The features of this simulation model include the ability to account for the experimentally observed stimulus-tuned adaptive temporal behaviour of time constants in the fly visual system. The simulation results obtained, clearly show that the application of such an adaptive processing procedure delivers an improved imaging technique of moving patterns in the high velocity range.

## A FEW REMARKS ON THE FLY VISUAL SYSTEM

The visual system of the diptera, including the blowfly *Calliphora erythrocephala (Mg.)* is very regularly organized and allows therefore very precise optical stimulation techniques. Also, long term electrophysiological recordings can be made relatively easy in this visual system. For these reasons the blowfly (which is well-known as a very rapid and 'clever' pilot) turns out to be an extremely suitable animal for a systematic study of basic principles that may underlie the detection and further processing of movement information at the neural level.

In the fly visual system the input retinal mosaic structure is precisely mapped onto the higher order optic ganglia (lamina, medulla, lobula). This means that each neural column in each ganglion in this visual system corresponds to a certain optical axis in the visual field of the compound eye. In the lobula complex a set of wide-field movement sensitive neurons is found, each of which integrates the input signals over the whole visual field of the entire eye. One of these wide field neurons, that has been classified as H1 by Hausen[1] has been extensively studied both anatomically[2, 3, 4] as well as electrophysiologically[5, 6, 7]. The obtained results generally agree very well with those found in behavioral optomotor experiments on movement detection[8] and can be understood in terms of Reichardts correlation model[9, 10].

The H1 neuron is sensitive to horizontal movement and directionally selective: very high rates of action potentials (*spikes*) up to 300 per second can be recorded from this element in the case of visual stimuli which move horizontally inward, i.e. from back to front in the visual field (*preferred direction*), whereas movement horizontally outward, i.e. from front to back (*null direction*) suppresses its activity.

# EXPERIMENTAL RESULTS AS A MODELLING BASE

When the H1 neuron is stimulated in its preferred direction with a step wise pattern displacement, it will respond with an increase of neural activity. By repeating this stimulus step over and over one can obtain the averaged response: after a 20 ms latency period the response manifests itself as a sharp increase in average firing rate followed by a much slower decay to the spontaneous activity level. Two examples of such averaged responses are shown in the Post Stimulus Time Histograms (PSTH's) of figure 1. Time to peak and peak height are related and depend on modulation depth, stimulus step size and spatial extent of the stimulus. The tail of the responses can be described adequately by an exponential decay toward a constant spontaneous firing rate:

$$R(t)=c+a \cdot e^{(-t/\tau)} \tag{1}$$

For each setting of the stimulus parameters, the response parameters, defined by equation (1), can be estimated by a least-squares fit to the tail of the PSTH. The smooth lines in figure 1 are the results of two such fits.

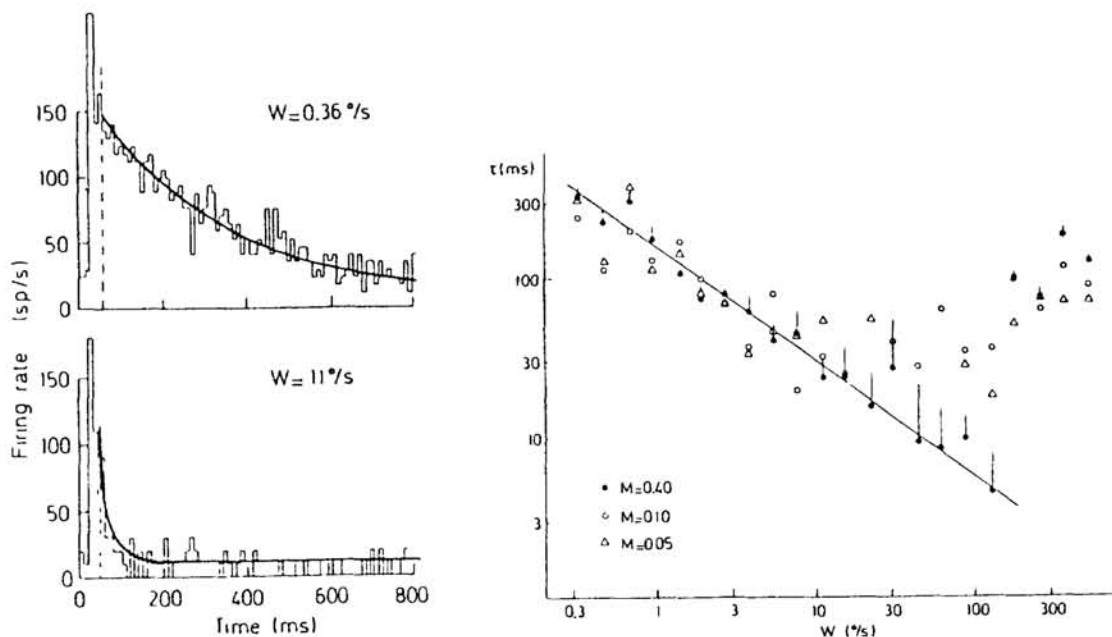

Fig.1    Averaged responses (PSTH's) obtained from the H1 neuron, being adapted to smooth stimulus motion with velocities 0.36°/s (top) and 11°/s (bottom) respectively. The smooth lines represent least-squares fits to the PSTH's of the form $R(t)=c+a \cdot e^{(-t/\tau)}$. Values of $\tau$ for the two PSTH's are 331 and 24 ms respectively (de Ruyter van Steveninck et al.[7]).

Fig.2    Fitted values of $\tau$ as a function of adaptation velocity for three modulation depths M. The straight line is a least-squares fit to represent the data for M=0.40 in the region w=0.3-100°/s. It has the form $\tau = \alpha \cdot w^{-\beta}$ with $\alpha$=150 ms and $\beta$=0.7 (de Ruyter van Steveninck et al.[7]).

Figure 2 shows fitted values of the response time constant $\tau$ as a function of the angular velocity of a moving stimulus (a square wave grating in most experiments) which was presented to the animal during a period long enough to let its visual system adapt to this moving pattern and before the step wise pattern displacement (which reveals $\tau$) was given. The straight line, described by

$$\tau = \alpha \cdot W^{-\beta} \tag{2}$$

(with W in °/s and $\tau$ in ms) represents a least-squares fit to the data over the velocity range from 0.36 to 125 °/s. For this range, $\tau$ varies from 320 to roughly 10 ms, with $\alpha=150\pm10$ ms and $\beta=0.7\pm0.05$. Defining the adaptation range of $\tau$ as that interval of velocities for which $\tau$ decreases with increasing velocity, we may conclude from figure 2 that within the adaptation range, $\tau$ is not very sensitive to the modulation depth.

The outcome of similar experiments with a constant modulation depth of the pattern (M=0.40) and a constant pattern velocity but with four different values of the contrast frequency $f_c$ (i.e. the number of spatial periods per second that traverse an individual visual axis as determined by the spatial wavelength $\lambda_s$ of the pattern and the pattern velocity v according to $f_c=v/\lambda_s$) reveal also an almost complete independency of the behaviour of $\tau$ on contrast frequency. Other experiments in which the stimulus field was subdivided into regions with different adaptation velocities, made clear that the time constants of the input channels of the H1 neuron were set locally by the values of the stimulus velocity in each stimulus sub-region. Finally, it was found that the adaptation of $\tau$ is driven by the stimulus velocity, independent of its direction.

These findings can be summarized qualitatively as follows: in steady state, the response time constants $\tau$ of the neural units at the highest level in the fly visual system are found to be tuned locally within a large velocity range exclusively by the magnitude of the velocity of the moving pattern and not by its direction, despite the directional selectivity of the neuron itself. We will not go into the question of how this amazing adaptive mechanism may be hard-wired in the fly visual system. Instead we will make advantage of the results derived thus far and attempt to fit the experimental observations into an image processing approach. A large number of theories and several distinct classes of algorithms to encode velocity and direction of movement in visual systems have been suggested by, for example, Marr and Ullman[11] and van Santen and Sperling[12].

We hypothesize that the adaptive mechanism for the setting of the time constants leads to an optimization for the overall performance of the visual system by realizing a velocity independent representation of the moving object. In other words: within the range of velocities for which the time constants are found to be tuned by the velocity, the representation of that stimulus at a certain level within the visual circuitry, should remain independent of any variation in stimulus velocity.

## OBJECT MOTION DEGRADATION: MODELLING

Given the physical description of motion and a linear space invariant model, the motion degradation process can be represented by the following convolution integral:

$$g(x,y)=\int_{-\infty}^{\infty} \int_{-\infty}^{\infty} (h(x-u,y-v) \cdot f(u,v))\, dudv \tag{3}$$

where f(u,v) is the object intensity at position (u,v) in the object coordinate frame, h(x-u,y-v) is the Point Spread Function (PSF) of the imaging system, which is the response at (x,y) to a unit pulse at (u,v) and g(x,y) is the image intensity at the spatial position (x,y) as blurred by the imaging system. Any possible additive white noise degradation of the already motion blurred image is neglected in the present considerations.

For a review of principles and techniques in the field of digital image degradation and restoration, the reader is referred to Harris[13], Sawchuk[14], Sondhi[15], Nahi[16], Aboutalib et al.[17, 18], Hildebrand[19], Rajala de Figueiredo[20]. It has been demonstrated first by Aboutalib et al.[17] that for situations in which the motion blur occurs in a straight line along one spatial coordinate, say along the horizontal axis, it is correct to look at the blurred image as a collection of degraded line scans through the entire image. The dependence on the vertical coordinate may then be dropped and eq. (3) reduces to:

$$g(x)= \int_{-\infty}^{\infty} h(x-u) \cdot f(u)du \qquad (4)$$

Given the mathematical description of the relative movement, the corresponding PSF can be derived exactly and equation (4) becomes:

$$g(x)= \int_{R} h(x-u) \cdot f(u)du \qquad (5)$$

where R is the extent of the motion blur. Typically, a discrete version of (5), applicable for digital image processing purposes, is described by:

$$g(k)=\sum_{1}^{L} h(k-l) \cdot f(l) \qquad ; k=1,...,N \qquad (6)$$

where k and l take on integer values and L is related to the motion blur extent.

According to Aboutalib et al.[18] a scalar difference equation model (M,a,b,c) can then be derived to model the motion degradation process:

$$x(k+1) = M \cdot x(k)+a \cdot f(k)$$

$$g(k) = b \cdot x(k)+c \cdot f(k) \qquad ; k=1,...,N \qquad (7)$$

$$h(i) = c_0\Delta(i)+c_1\Delta(i-1)+ ...... +c_m\Delta(i-m)$$

where x(k) is the m-dimensional state vector at position k along a scan line, f(k) is the input intensity at position k, g(k) is the output intensity, m is the blur extent, N is the number of elements in a line, c is a scalar, M, a and b are constant matrices of order (mxm), (mx1) and (1xm) respectively, containing the discrete values $c_j$ of the blurring PSF h(j) for j=0,...,m and $\Delta(.)$ is the Kronecker delta function.

# INFLUENCE OF BOTH TIME CONSTANT AND VELOCITY ON THE AMOUNT OF MOTION BLUR IN AN ARTIFICIAL RECEPTOR ARRAY

To start with, we incorporate in our simulation model a PSF, derived from equation (1), to model the performance of all neural columnar arranged filters in the lobula complex, with the restriction that the time constants $\tau$ remain fixed throughout the whole range of stimulus velocities. Realization of this PSF can easily be achieved via the just mentioned state space model.

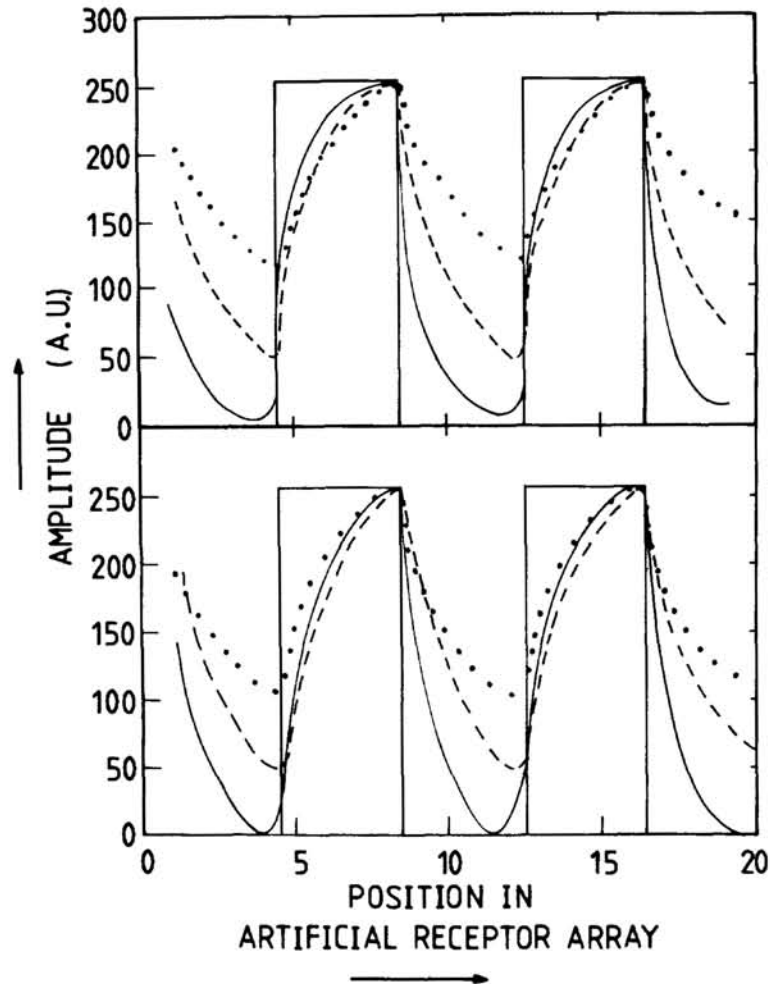

Fig.3    **upper part.** Demonstration of the effect that an increase in magnitude of the time constants of an one-dimensional array of filters will result in increase in motion blur (while the pattern velocity remains constant). Original pattern shown in solid lines is a square-wave grating with a spatial wavelength equal to 8 artificial receptor distances. The three other wave forms drawn, show that for a gradual increase increase in magnitude of the time constants, the representation of the original square-wave will consequently degrade. **lower part.** A gradual increase in velocity of the moving square-wave (while the filter time constants are kept fixed) results also in a clear increase of degradation.

First we demonstrate the effect that an increase in time constant (while the pattern velocity remains the same) will result in an increase in blur. Therefore we introduce an one dimensional array of filters all being equipped with the same time constant in their impulse response. The original pattern shown in square and solid lines in the upper part of figure 3 consists of a square wave grating with a spatial period overlapping 8 artificial receptive filters. The 3 other patterns drawn there show that for the same constant velocity of the moving grating, an increase in the magnitude of the time constants of the filters results in an increased blur in the representation of that grating. On the other hand, an increase in velocity (while the time constants of the artificial receptive units remain the same) also results in a clear increase in motion blur, as demonstrated in the lower part of figure 3.

Inspection of the two wave forms drawn by means of the dashed lines in both upper and lower half of the figure, yields the conclusion, that (apart from rounding errors introduced by the rather small number of artificial filters available), equal amounts of smear will be produced when the product of time constant and pattern velocity is equal. For the upper dashed wave form the velocity was four times smaller but the time constant four times larger than for its equivalent in the lower part of the figure.

## ADAPTIVE SCHEME

In designing a proper image processing procedure our next step is to incorporate the experimentally observed flexibility property of the time constants in the imaging elements of our device. In figure 4$^a$ a scheme is shown, which filters the information with fixed time constants, not influenced by the pattern velocity. In figure 4$^b$ a network is shown where the time constants also remain fixed no matter what pattern movement is presented, but now at the next level of information processing, a spatially differential network is incorporated in order to enhance blurred contrasts.

In the filtering network in figure 4$^c$, first a measurement of the magnitude of the velocity of the moving objects is done by thus far hypothetically introduced movement processing algorithms, modelled here as a set of receptive elements sampling the environment in such a manner that proper estimation of local pattern velocities can be done. Then the time constants of the artificial receptive elements will be tuned according to the estimated velocities and finally the same differential network as in scheme 4$^b$, is used.

The actual tuning mechanism used for our simulations is outlined in figure 5: once given the range of velocities for which the model is supposed to be operational, and given a lower limit for the time constant $\tau_{min}$ ($\tau_{min}$ can be the smallest value which physically can be realized), the time constant will be tuned to a new value according to the experimentally observed reciprocal relationship, and will, for all velocities within the adaptive range, be larger than the fixed minimum value. As demonstrated in the previous section the corresponding blur in the representation of the moving stimulus will thus always be larger than for the situation in which the filtering is done with fixed and smallest time constants $\tau_{min}$. More important however is the fact that due to this tuning mechanism the blur will be constant since the product of velocity and time constant is kept constant. So, once the information has been processed by such a system, a velocity independent representation of the image will be the result, which can serve as the input for the spatially differentiating network as outlined in figure 4$^c$.

The most elementary form for this differential filtering procedure is the one

in which the gradient of two filters K-1 and K+1 which are the nearest neighbors of filter K, is taken and then added with a constant weighing factor to the central output K as drawn in figure 4$^b$ and 4$^c$, where the sign of the gradient depends on the direction of the estimated movement. Essential for our model is that we claim that this weighing factor should be constant throughout the whole set of filters and for the whole high velocity range in which the heterodyne imaging has to be performed. Important to notice is the existence of a so-called settling time, i.e. the minimal time needed for our movement processing device to be able to accurately measure the object velocity. [Note: this time can be set equal to zero in the case that the relative stimulus velocity is known a priori, as demonstrated in figure 3]. Since, without doubt, within this settling period estimated velocity values will come out erroneously and thus no optimal performance of our imaging device can be expected, in all further examples, results after this initial settling procedure will be shown.

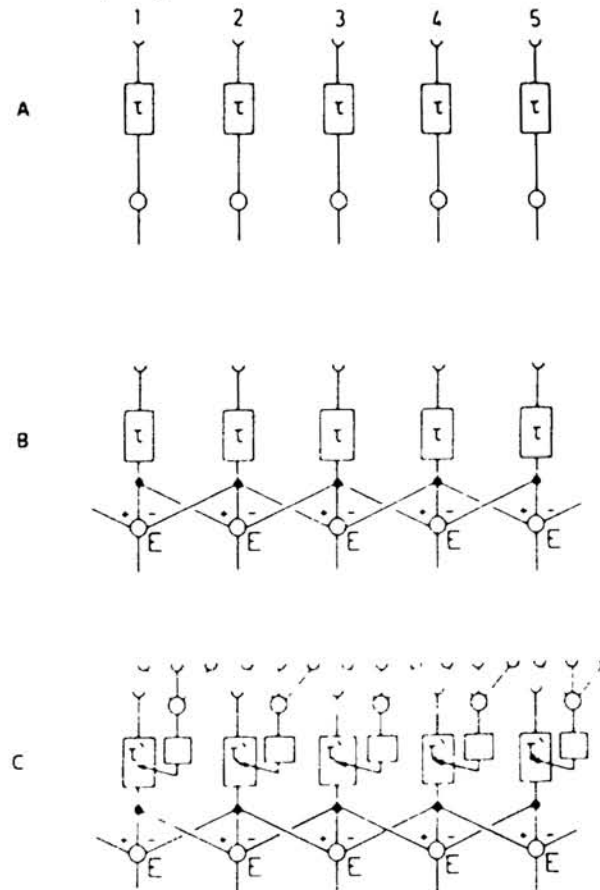

Fig. 4    Pattern movement in this figure is to the right.

A: Network consisting of a set of filters with a fixed, pattern velocity independent, time constant in their impulse response.

B: Identical network as in figure 4A now followed by a spatially differentiating circuitry which adds the weighed gradients of two neighboring filter outputs K-1 and K+1 to the central filter output K.

C: The time constants of the filtering network are tuned by a hypothetical movement estimating mechanism, visualized here as a number of receptive elements, of which the combined output tunes the filters. A detailed description of this mechanism is shown in figure 5. This tuned network is followed by an identical spatially differentiating circuit as described in figure 4B.

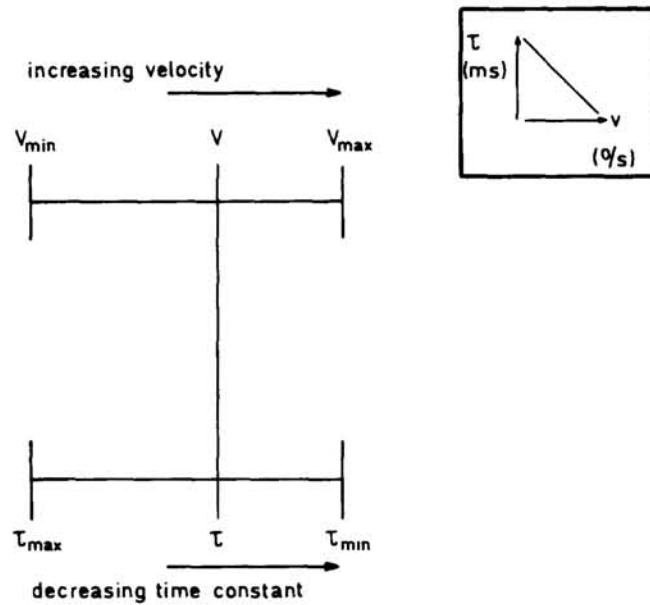

Fig. 5    Detailed description of the mechanism used to tune the time constants. The time constant $\tau$ of a specific neural channel is set by the pattern velocity according to the relationship shown in the insert, which is derived from eq. (2) with $\alpha=1$ and $\beta=1$.

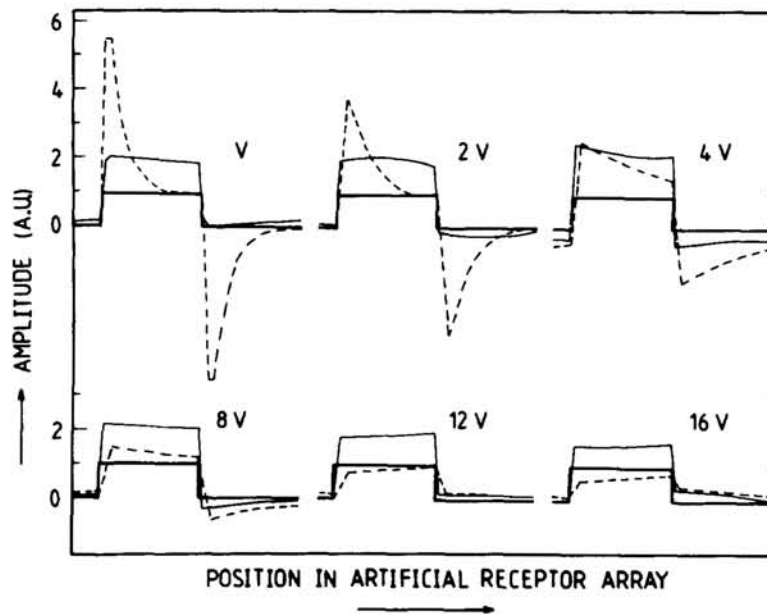

Fig.6    **Thick lines:** square-wave stimulus pattern with a spatial wavelength overlapping 32 artificial receptive elements. **Thick lines:** responses for 6 different pattern velocities in a system consisting of paralleling neural filters equipped with time constants, tuned by this velocity, and followed by a spatially differentiating network as described.
**Dashed lines:** responses to the 6 different pattern velocities in a filtering system with fixed time constants, followed by the same spatial differentiating circuitry as before. Note the sharp over- and under shoots for this case.

Results obtained with an imaging procedure as drawn in figure 4[b] and 4[c] are shown in figure 6. The pattern consists of a square wave, overlapping 32 picture elements. The pattern moves (to the left) with 6 different velocities v, 2v, 4v, 8v, 12v, 16v. At each velocity only one wavelength is shown. Thick lines: square wave pattern. Dashed lines: the outputs of an imaging device as depicted in figure 4[b]: constant time constants and a constant weighing factor in the spatial processing stage. Note the large differences between the several outputs. Thin continuous lines: the outputs of an imaging device as drawn in figure 4[c]: tuned time constants according to the reciprocal relationship between pattern velocity and time constant and a constant weighing factor in the spatial processing stage. For further simulation details the reader is referred to Zaagman et al.[21]. Now the outputs are almost completely the same and in good agreement with the original stimulus throughout the whole velocity range.

Figure 7 shows the effect of the gradient weighing factor on the overall filter performance, estimated as the improvement of the deblurred images as compared with the blurred image, measured in dB. This quantitative measure has been determined for the case of a moving square wave pattern with motion blur

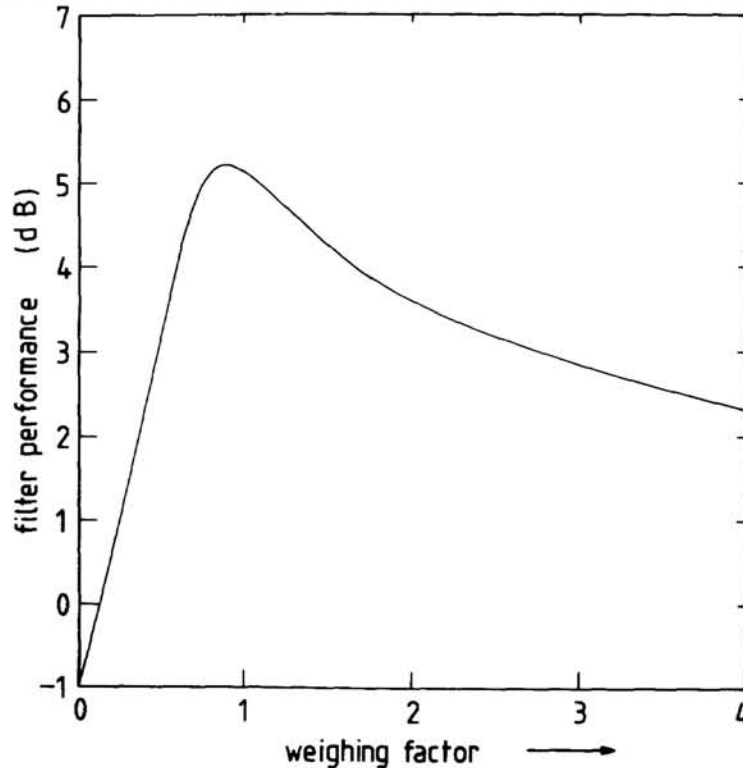

Fig. 7    Effect of the weighing factor on the overall filter performance. Curve measured for the case of a moving square-wave grating. Filter performance is estimated as the improvement in signal to noise ratio:

$$I = 10 \cdot {}^{10}\log \left( \frac{\Sigma_i \Sigma_j((v(i,j)-u(i,j))^2}{\Sigma_i \Sigma_j((\hat{u}(i,j)-u(i,j))^2} \right)$$

where $u(i,j)$ is the original intensity at position $(i,j)$ in the image, $v(i,j)$ is the intensity at the same position $(i,j)$ in the motion blurred image and $\hat{u}(i,j)$ is the intensity at $(i,j)$ in the image, generated with the adaptive tuning procedure.

extents comparable to those used for the simulations to be discussed in section IV. From this curve it is apparent that for this situation there is an optimum value for this weighing factor. Keeping the weight close to this optimum value will result in a constant output of our adaptive scheme, thus enabling an optimal deblurring of the smeared image of the moving object.

On the other hand, starting from the point of view that the time constants should remain fixed throughout the filtering process, we should had have to tune the gradient weights to the velocity in order to produce a constant output as demonstrated in figure 6 where the dashed lines show strongly differing outputs of a fixed time constant system with spatial processing with constant weight (figure 4$^b$). In other words, tuning of the time constants as proposed in this section results in: 1) the realization of the blur-constancy criterion as formulated previously, and 2) -as a consequence- the possibility to deblur the obtained image optimally with one and the same weighing factor of the gradient in the final spatial processing layer over the whole heterodyne velocity range.

## COMPUTER SIMULATION RESULTS AND CONCLUSIONS

The image quality improvement algorithm developed in the present contribution has been implemented on a general purpose DG Eclipse S/140 mini-computer for our two dimensional simulations. Figure 8$^a$ shows an undisturbed image, consisting of 256 lines of each 256 pixels, with 8 bit intensity resolution. Figure 8$^b$ shows what happens with the original image if the PSF is modelled according to the exponential decay (2). In this case the time constants of all spatial information processing channels have been kept fixed. Again, information content in the higher spatial frequencies has been reduced largely. The implementation of the heterodyne filtering procedure was now done as follows: first the adaptation range was defined by setting the range of velocities. This means that our adaptive heterodyne algorithm is supposed to operate adequately only within the thus defined velocity range and that -in that range- the time constants are tuned according to relationship (2) and will always come out larger than the minimum value $\tau_{min}$. For demonstration purposes we set $\alpha=1$ and $\beta=1$ in eq. (2), thus introducing the phenomenon that for any velocity, the two dimensional set of spatial filters with time constants tuned by that velocity, will always produce a constant output, independent of this velocity which introduces the motion blur. Figure 8$^c$ shows this representation. It is important to note here that this constant output has far more worse quality than any set of filters with smallest and fixed time constants $\tau_{min}$ would produce for velocities within the operational range. The advantage of a velocity independent output at this level in our simulation model, is that in the next stage a differential scheme can be implemented as discussed in detail in the preceding paragraph. Constancy of the weighing factor which is used in this differential processing scheme is guaranteed by the velocity independency of the obtained image representation.

Figure 8$^d$ shows the result of the differential operation with an optimized gradient weighing factor. This weighing factor has been optimized based on an almost identical performance curve as described previously in figure 7. A clear and good restoration is apparent from this figure, though close inspection reveals fine structure (especially for areas with high intensities) which is unrelated with the original intensity distribution. These artifacts are caused by the phenomenon that for these high intensity areas possible tuning errors will show up much more pronounced than for low intensities.

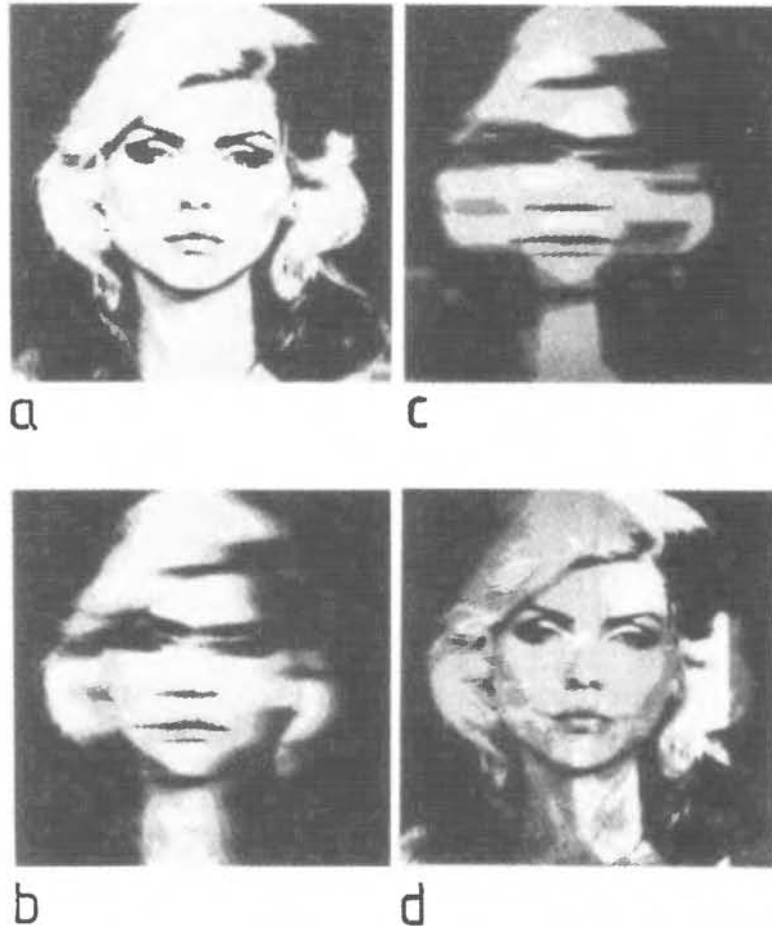

a    c

b    d

Fig. 8a    Original 256x256x8 bit picture.
Fig. 8b    Motion degraded image with a PSF derived from $R(t)=c+a\cdot e^{(-t/\tau)}$, where $\tau$ is kept fixed to 12 pixels and the motion blur extent is 32 pixels.
Fig. 8c    Worst case, i.e. the result of motion degradation of the original image with a PSF as in figure $8^b$, but with tuning of the time constants based on the velocity.
Fig. 8d    Restored version of the degraded image using the heterodyne adaptive processing scheme.

In conclusion: a heterodyne adaptive image processing technique, inspired by the fly visual system, has been presented as an imaging device for moving objects. A scalar difference equation model has been used to represent the motion blur degradation process. Based on the experimental results described and on this state space model, we developed an adaptive filtering scheme, which produces at a certain level within the system a constant output, permitting further differential operations in order to produce an optimally deblurred representation of the moving object.

## ACKNOWLEDGEMENTS

The authors wish to thank mr. Eric Bosman for his expert programming

assistance, mr. Franco Tommasi for many inspiring discussions and advises during the implementation of the simulation model and dr. Rob de Ruyter van Steveninck for experimental help. This research was partly supported by the *Netherlands Organization for the Advancement of Pure Research (Z.W.O.)* through the foundation *Stichting voor Biofysica.*

# REFERENCES

1. K. Hausen, Z. Naturforschung **31c**, 629-633 (1976).
2. N. J. Strausfeld, Atlas of an insect brain (Springer Verlag, Berlin, Heidelberg, New York, 1976).
3. K. Hausen, Biol. Cybern. **45**, 143-156 (1982).
4. R. Hengstenberg, J. Comp. Physiol. **149**, 179-193 (1982).
5. W. H. Zaagman, H. A. K. Mastebroek, J. W. Kuiper, Biol. Cybern. **31**, 163-168 (1978).
6. H. A. K. Mastebroek, W. H. Zaagman, B. P. M. Lenting, Vision Res. **20**, 467-474 (1980)
7. R. R. de Ruyter van Steveninck, W. H. Zaagman, H. A. K. Mastebroek, Biol. Cybern., **54**, 223-236 (1986).
8. W. Reichardt, T. Poggio, Q. Rev. Biophys. **9**, 311-377 (1976).
9. W. Reichardt, *in* Reichardt, W. (Ed.) Processing of optical Data by Organisms and Machines (Academic Press, New York, 1969), pp. 465-493.
10. T. Poggio, W. Reichardt, Q. Rev. Bioph. **9**, 377-439 (1976).
11. D. Marr, S. Ullman, Proc. R. Soc. Lond. **211**, 151-180 (1981).
12. J. P. van Santen, G. Sperling, J. Opt. Soc. Am. A **1**, 451-473 (1984).
13. J. L. Harris SR., J. Opt. Soc. Am. **56**, 569-574 (1966).
14. A. A. Sawchuk, Proc. IEEE, Vol. 60, No. 7, 854-861 (1972).
15. M. M.Sondhi, Proc. IEEE, Vol. 60, No. 7, 842-853 (1972).
16. N. E. Nahi, Proc. IEEE, Vol. 60, No. 7, 872-877 (1972).
17. A. O. Aboutalib, L. M. Silverman, IEEE Trans. On Circuits And Systems T-CAS **75**, 278-286 (1975).
18. A. O. Aboutalib, M. S. Murphy, L.M. Silverman, IEEE Trans. Automat. Contr. AC **22**, 294-302 (1977).
19. Th. Hildebrand, Biol. Cybern. **36**, 229-234 (1980).
20. S. A. Rajala, R. J. P. de Figueiredo, IEEE Trans. On Acoustics, Speech and Signal Processing, Vol. ASSSP-29, No. 5, 1033-1042 (1981).
21. W. H. Zaagman, H. A. K. Mastebroek, R. R. de Ruyter van Steveninck, IEEE Trans, Syst. Man Cybern. SMC **13**, 900-906 (1983).
